# EM Optimization of Latent-Variable Density Models

**Christopher M Bishop, Markus Svensén and Christopher K I Williams**
Neural Computing Research Group
Aston University, Birmingham, B4 7ET, UK
c.m.bishop@aston.ac.uk svensjfm@aston.ac.uk c.k.i.williams@aston.ac.uk

## Abstract

There is currently considerable interest in developing general non-linear density models based on latent, or hidden, variables. Such models have the ability to discover the presence of a relatively small number of underlying 'causes' which, acting in combination, give rise to the apparent complexity of the observed data set. Unfortunately, to train such models generally requires large computational effort. In this paper we introduce a novel latent variable algorithm which retains the general non-linear capabilities of previous models but which uses a training procedure based on the EM algorithm. We demonstrate the performance of the model on a toy problem and on data from flow diagnostics for a multi-phase oil pipeline.

## 1 INTRODUCTION

Many conventional approaches to density estimation, such as mixture models, rely on linear superpositions of basis functions to represent the data density. Such approaches are unable to discover structure within the data whereby a relatively small number of 'causes' act in combination to account for apparent complexity in the data. There is therefore considerable interest in *latent variable* models in which the density function is expressed in terms of of hidden variables. These include density networks (MacKay, 1995) and Helmholtz machines (Dayan *et al.*, 1995). Much of this work has been concerned with predicting binary variables. In this paper we focus on continuous data.

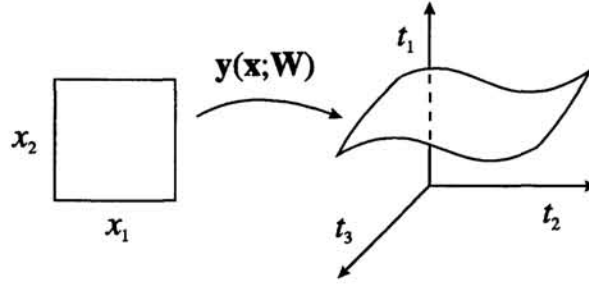

Figure 1: The latent variable density model constructs a distribution function in t-space in terms of a non-linear mapping $\mathbf{y}(\mathbf{x}; \mathbf{W})$ from a latent variable x-space.

## 2    THE LATENT VARIABLE MODEL

Suppose we wish to model the distribution of data which lives in a $D$-dimensional space $\mathbf{t} = (t_1, \ldots, t_D)$. We first introduce a transformation from the hidden variable space $\mathbf{x} = (x_1, \ldots, x_L)$ to the data space, governed by a non-linear function $\mathbf{y}(\mathbf{x}; \mathbf{W})$ which is parametrized by a matrix of weight parameters $\mathbf{W}$. Typically we are interested in the situation in which the dimensionality $L$ of the latent variable space is less than the dimensionality $D$ of the data space, since we wish to capture the fact that the data itself has an intrinsic dimensionality which is less than $D$. The transformation $\mathbf{y}(\mathbf{x}; \mathbf{W})$ then maps the hidden variable space into an $L$-dimensional non-Euclidean subspace embedded within the data space. This is illustrated schematically for the case of $L = 2$ and $D = 3$ in Figure 1.

If we define a probability distribution $p(\mathbf{x})$ on the latent variable space, this will induce a corresponding distribution $p(\mathbf{y})$ in the data space. We shall refer to $p(\mathbf{x})$ as the prior distribution of $\mathbf{x}$ for reasons which will become clear shortly. Since $L < D$, the distribution in t-space would be confined to a manifold of dimension $L$ and hence would be singular. Since in reality data will only approximately live on a lower-dimensional space, it is appropriate to include a noise model for the $\mathbf{t}$ vector. We therefore define the distribution of $\mathbf{t}$, for given $\mathbf{x}$ and $\mathbf{W}$, given by a spherical Gaussian centred on $\mathbf{y}(\mathbf{x}; \mathbf{W})$ having variance $\beta^{-1}$ so that

$$p(\mathbf{t}|\mathbf{x}, \mathbf{W}) = \left(\frac{\beta}{2\pi}\right)^{D/2} \exp\left\{-\frac{\beta}{2}\sum_{k=1}^{D}\{y_k(\mathbf{x}; \mathbf{W}) - t_k\}^2\right\}. \tag{1}$$

The distribution in t-space, for a given value of the weight matrix $\mathbf{W}$, is then obtained by integration over the x-distribution

$$p(\mathbf{t}|\mathbf{W}) = \int p(\mathbf{t}|\mathbf{x}, \mathbf{W})p(\mathbf{x})\, d\mathbf{x}. \tag{2}$$

For a given data set $\mathcal{D} = (\mathbf{t}^1, \ldots, \mathbf{t}^N)$ of $N$ data points, we can determine the weight matrix $\mathbf{W}$ using maximum likelihood. For convenience we introduce an error function given by the negative log likelihood:

$$E(\mathbf{W}) = -\ln\prod_{n=1}^{N}p(\mathbf{t}^n|\mathbf{W}) = -\sum_{n=1}^{N}\ln\left\{\int p(\mathbf{t}^n|\mathbf{x}^n, \mathbf{W})p(\mathbf{x}^n)\, d\mathbf{x}^n\right\}. \tag{3}$$

In principle we can now seek the maximum likelihood solution for the weight matrix, once we have specified the prior distribution $p(\mathbf{x})$ and the functional form of the mapping $\mathbf{y}(\mathbf{x}; \mathbf{W})$, by minimizing $E(\mathbf{W})$. However, the integrals over $\mathbf{x}$ occuring in (3), and in the corresponding expression for $\nabla E$, will, in general, be analytically intractable. MacKay (1995) uses Monte Carlo techniques to evaluate these integrals and conjugate gradients to find the weights. This is computationally very intensive, however, since a Monte Carlo integration must be performed every time the conjugate gradient algorithm requests a value for $E(\mathbf{W})$ or $\nabla E(\mathbf{W})$. We now show how, by a suitable choice of model, it is possible to find an EM algorithm for determining the weights.

## 2.1   EM ALGORITHM

There are three key steps to finding a tractable EM algorithm for evaluating the weights. The first is to use a generalized linear network model for the mapping function $\mathbf{y}(\mathbf{x}; \mathbf{W})$. Thus we write

$$\mathbf{y}(\mathbf{x}; \mathbf{W}) = \mathbf{W}\phi(\mathbf{x}) \tag{4}$$

where the elements of $\phi(\mathbf{x})$ consist of $M$ fixed basis functions $\phi_j(\mathbf{x})$, and $\mathbf{W}$ is a $D \times M$ matrix with elements $w_{kj}$. Generalized linear networks possess the same universal approximation capabilities as multi-layer adaptive networks. The price which has to be paid, however, is that the number of basis functions must typically grow exponentially with the dimensionality $L$ of the input space. In the present context this is not a serious problem since the dimensionality is governed by the latent variable space and will typically be small. In fact we are particularly interested in visualization applications, for which $L = 2$.

The second important step is to use a simple Monte Carlo approximation for the integrals over $\mathbf{x}$. In general, for a function $Q(\mathbf{x})$ we can write

$$\int Q(\mathbf{x})p(\mathbf{x})\,d\mathbf{x} \simeq \frac{1}{K}\sum_{i=1}^{K} Q(\mathbf{x}^i) \tag{5}$$

where $\mathbf{x}^i$ represents a sample drawn from the distribution $p(\mathbf{x})$. If we apply this to (3) we obtain

$$E(\mathbf{W}) = -\sum_{n=1}^{N} \ln\left\{\frac{1}{K}\sum_{i=1}^{K} p(\mathbf{t}^n | \mathbf{x}^{ni}, \mathbf{W})\right\} \tag{6}$$

The third key step to choose the sample of points $\{\mathbf{x}^{ni}\}$ to be the same for each term in the summation over $n$. Thus we can drop the index $n$ on $\mathbf{x}^{ni}$ to give

$$E(\mathbf{W}) = -\sum_{n=1}^{N} \ln\left\{\frac{1}{K}\sum_{i=1}^{K} p(\mathbf{t}^n | \mathbf{x}^i, \mathbf{W})\right\} \tag{7}$$

We now note that (7) represents the negative log likelihood under a distribution consisting of a mixture of $K$ kernel functions. This allows us to apply the EM algorithm to find the maximum likelihood solution for the weights. Furthermore, as a consequence of our choice (4) for the non-linear mapping function, it will turn out that the M-step can be performed explicitly, leading to a solution in terms of a set

of linear equations. We note that this model corresponds to a *constrained* Gaussian mixture distribution of the kind discussed in Hinton *et al.* (1992).

We can formulate the EM algorithm for this system as follows. Setting the derivatives of (7) with respect to $w_{kj}$ to zero we obtain

$$\sum_{n=1}^{N}\sum_{i=1}^{K} R_{in}(\mathbf{W}) \left\{ \sum_{r=1}^{M} w_{kr}\phi_r(\mathbf{x}^i) - t_k^n \right\} \phi_j(\mathbf{x}^i) = 0 \tag{8}$$

where we have used Bayes' theorem to introduce the posterior probabilities, or *responsibilities*, for the mixture components given by

$$R_{in}(\mathbf{W}) = \frac{p(\mathbf{t}^n | \mathbf{x}^i, \mathbf{W})}{\sum_{i'=1}^{K} p(\mathbf{t}^n | \mathbf{x}^{i'}, \mathbf{W})}. \tag{9}$$

Similarly, maximizing with respect to $\beta$ we obtain

$$\frac{1}{\beta} = \frac{1}{ND} \sum_{i=1}^{K}\sum_{n=1}^{N} R_{ni}(\mathbf{W}) \left\| \mathbf{y}(\mathbf{x}^n; \mathbf{W}) - \mathbf{t}^n \right\|^2 . \tag{10}$$

The EM algorithm is obtained by supposing that, at some point in the algorithm, the current weight matrix is given by $\mathbf{W}^{\text{old}}$ and the current value of $\beta$ is $\beta^{\text{old}}$. Then we can evaluate the responsibilities using these values for $\mathbf{W}$ and $\beta$ (the E-step), and then solve (8) for the weights to give $\mathbf{W}^{\text{new}}$ and subsequently solve (10) to give $\beta^{\text{new}}$ (the M-step). The two steps are repeated until a suitable convergence criterion is reached. In practice the algorithm converges after a relatively small number of iterations.

A more formal justification for the EM algorithm can be given by introducing auxiliary variables to label which component is responsible for generating each data point, and then computing the expectation with respect to the distribution of these variables. Application of Jensen's inequality then shows that, at each iteration of the algorithm, the error function will decrease unless it is already at a (local) minimum, as discussed for example in Bishop (1995).

If desired, a regularization term can be added to the error function to control the complexity of the model $\mathbf{y}(\mathbf{x}; \mathbf{W})$. From a Bayesian viewpoint, this corresponds to a prior distribution over weights. For a regularizer which is a quadratic function of the weight parameters, this leads to a straightforward modification to the weight update equations. It is convenient to write the condition (8) in matrix notation as

$$(\mathbf{\Phi}^{\text{T}}\mathbf{G}^{\text{old}}\mathbf{\Phi} + \lambda\mathbf{I})(\mathbf{W}^{\text{new}})^{\text{T}} = \mathbf{\Phi}^{\text{T}}\mathbf{T}^{\text{old}} \tag{11}$$

where we have included a regularization term with coefficient $\lambda$, and $\mathbf{I}$ denotes the unit matrix. In (11) $\mathbf{\Phi}$ is a $K \times M$ matrix with elements $\Phi_{ij} = \phi_j(\mathbf{x}^i)$, $\mathbf{T}$ is a $K \times D$ matrix, and $\mathbf{G}$ is a $K \times K$ diagonal matrix, with elements

$$T_{ik} = \sum_{n=1}^{N} R_{in}(\mathbf{W}) t_k^n \qquad G_{ii} = \sum_{n=1}^{N} R_{in}(\mathbf{W}). \tag{12}$$

We can now solve (11) for $\mathbf{W}^{\text{new}}$ using standard linear matrix inversion techniques, based on singular value decomposition to allow for possible ill-conditioning. Note that the matrix $\mathbf{\Phi}$ is constant throughout the algorithm, and so need only be evaluated once at the start.

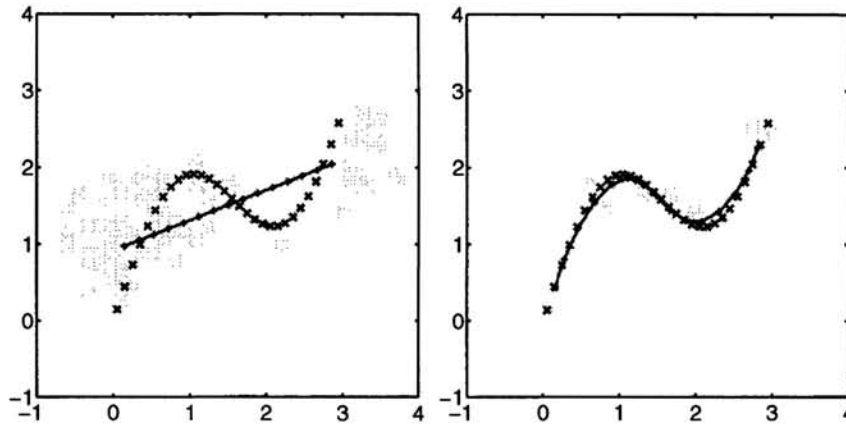

Figure 2: Results from a toy problem involving data ('×') generated from a 1-dimensional curve embedded in 2 dimensions, together with the projected sample points ('+') and their Gaussian noise distributions (filled circles). The initial configuration, determined by principal component analysis, is shown on the left, and an intermediate configuration, obtained after 4 iterations of EM, is shown on the right.

## 3   RESULTS

We now present results from the application of this algorithm first to a toy problem involving data in three dimensions, and then to a more realistic problem involving 12-dimensional data arising from diagnostic measurements of oil flows along multiphase pipelines.

For simplicity we choose the distribution $p(\mathbf{x})$ to be uniform over the unit square. The basis functions $\phi_j(\mathbf{x})$ are taken to be spherically symmetric Gaussian functions whose centres are distributed on a uniform grid in $\mathbf{x}$-space, with a common width parameter chosen so that the standard deviation is equal to the separation of neighbouring basis functions. For both problems the weights in the network were initialized by performing principal components analysis on the data and then finding the least-squares solution for the weights which best approximates the linear transformation which maps latent space to target space while generating the correct mean and variance in target space.

As a simple demonstration of this algorithm, we consider data generated from a one-dimensional distribution embedded in two dimensions, as shown in Figure 2.

### 3.1   OIL FLOW DATA

Our second example arises in the problem of determining the fraction of oil in a multi-phase pipeline carrying a mixture of oil, water and gas (Bishop and James, 1993). Each data point consists of 12 measurements taken from dual-energy gamma densitometers measuring the attenuation of gamma beams passing through the pipe. Synthetically generated data is used which models accurately the attenuation processes in the pipe, as well as the presence of noise (arising from photon statistics). The three phases in the pipe (oil, water and gas) can belong to one of three different geometrical configurations, corresponding to stratified, homogeneous, and annular flows, and the data set consists of 1000 points distributed equally between the 3

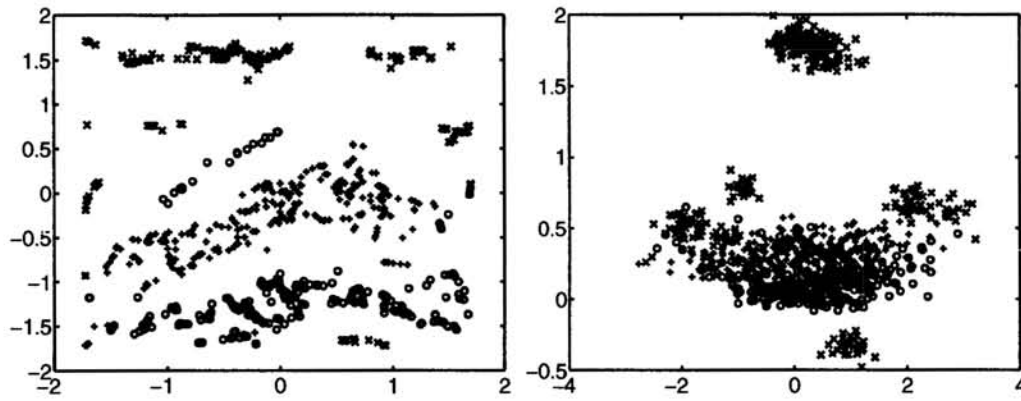

Figure 3: The left plot shows the posterior-mean projection of the oil data in the latent space of the non-linear model. The plot on the right shows the same data set projected onto the first two principal components. In both plots, crosses, circles and plus-signs represent the stratified, annular and homogeneous configurations respectively.

classes. We take the latent variable space to be two-dimensional. This is appropriate for this problem as we know that, locally, the data must have an intrinsic dimensionality of two (neglecting noise on the data) since, for any given geometrical configuration of the three phases, there are two degrees of freedom corresponding to the fractions of oil and water in the pipe (the fraction of gas being redundant since the three fractions must sum to one). It also allows us to use the latent variable model to visualize the data by projection onto x-space.

For the purposes of visualization, we note that a data point $t^n$ induces a posterior distribution $p(x|t^n, W^*)$ in x-space, where $W^*$ denotes the value of the weight matrix for the trained network. This provides considerably more information in the visualization space than many simple techniques (which generally project each data point onto a single point in the visualization space). For example, the posterior distribution may be multi-modal, indicating that there is more than one region of x-space which can claim significant responsibility for generating the data point. However, it is often convenient to project each data point down to a unique point in x-space. This can be done by finding the mean of the posterior distribution, which itself can be evaluated by a simple Monte Carlo integration using quantities already calculated in the evaluation of $W^*$.

Figure 3 shows the oil data visualized in the latent-variable space in which, for each data point, we have plotted the posterior mean vector. Again the points have been labelled according to their multi-phase configuration. We have compared these results with those from a number of conventional techniques including *factor analysis* and *principal component analysis*. Note that factor analysis is precisely the model which results if a linear mapping is assumed for $y(x; W)$, a Gaussian distribution $p(x)$ is chosen in the latent space, and the noise distribution in data space is taken to be Gaussian with a diagonal covariance matrix. Of these techniques, principal component analysis gave the best class separation (assessed subjectively) and is illustrated in Figure 3. Comparison with the results from the non-linear model clearly shows that the latter gives much better separation of the three classes, as a consequence of the non-linearity permitted by the latent variable mapping.

## 4 DISCUSSION

There are interesting relationships between the model discussed here and a number of well-known algorithms for unsupervised learning. We have already commented that factor analysis is a special case of this model, involving a linear mapping from latent space to data space. The Kohonen topographic map algorithm (Kohonen, 1995) can be regarded as an approximation to a latent variable density model of the kind outlined here. Finally, there are interesting similarities to a 'soft' version of the 'principal curves' algorithm (Tibshirani, 1992).

The model we have described can readily be extended to deal with the problem of missing data, provided we assume that the missing data is *ignorable* and *missing at random* (Little and Rubin, 1987). This involves maximizing the likelihood function in which the missing values have been integrated out. For the model discussed here, the integrations can be performed analytically, leading to a modified form of the EM algorithm.

Currently we are extending the model to allow for mixed continuous and categorical variables. We are also exploring Bayesian approaches, based on Markov chain Monte Carlo, to replace the maximum likelihood procedure.

### Acknowledgements

This work was partially supported by EPSRC grant GR/J75425: *Novel Developments in Learning Theory*. Markus Svensén would like to thank the staff of the SANS group in Stockholm for their hospitality during part of this project.

## References

Bishop, C. M. (1995). *Neural Networks for Pattern Recognition*. Oxford University Press.

Bishop, C. M. and G. D. James (1993). Analysis of multiphase flows using dual-energy gamma densitometry and neural networks. *Nuclear Instruments and Methods in Physics Research* **A327**, 580–593.

Dayan, P., G. E. Hinton, R. M. Neal, and R. S. Zemel (1995). The Helmholtz machine. *Neural Computation* **7** (5), 889–904.

Hinton, G. E., C. K. I. Williams, and M. D. Revow (1992). Adaptive elastic models for hand-printed character recognition. In J. E. Moody, S. J. Hanson, and R. P. Lippmann (Eds.), *Advances in Neural Information Processing Systems 4*. Morgan Kauffmann.

Kohonen, T. (1995). *Self-Organizing Maps*. Berlin: Springer-Verlag.

Little, R. J. A. and D. B. Rubin (1987). *Statistical Analysis with Missing Data*. New York: John Wiley.

MacKay, D. J. C. (1995). Bayesian neural networks and density networks. *Nuclear Instruments and Methods in Physics Research, A* **354** (1), 73–80.

Tibshirani, R. (1992). Principal curves revisited. *Statistics and Computing* **2**, 183–190.